# Interpreting the Neural Code with Formal Concept Analysis

**Dominik Endres, Peter Földiák**
School of Psychology,University of St. Andrews
KY16 9JP, UK
{dme2,pf2}@st-andrews.ac.uk

## Abstract

We propose a novel application of Formal Concept Analysis (FCA) to neural decoding: instead of just trying to figure out which stimulus was presented, we demonstrate how to explore the semantic relationships in the neural representation of large sets of stimuli. FCA provides a way of displaying and interpreting such relationships via concept lattices. We explore the effects of neural code sparsity on the lattice. We then analyze neurophysiological data from high-level visual cortical area STSa, using an exact Bayesian approach to construct the formal context needed by FCA. Prominent features of the resulting concept lattices are discussed, including hierarchical face representation and indications for a product-of-experts code in real neurons.

## 1   Introduction

Mammalian brains consist of billions of neurons, each capable of independent electrical activity. From an information-theoretic perspective, the patterns of activation of these neurons can be understood as the codewords comprising the neural code. The neural code describes which pattern of activity corresponds to what information item. We are interested in the (high-level) visual system, where such items may indicate the presence of a stimulus object or the value of some stimulus attribute, assuming that each time this item is represented the neural activity pattern will be the same or at least similar. *Neural decoding* is the attempt to reconstruct the stimulus from the observed pattern of activation in a given population of neurons [1, 2, 3, 4]. Popular decoding quality measures, such as Fisher's linear discriminant [5] or mutual information [6] capture how accurately a stimulus can be determined from a neural activity pattern (e.g., [4]). While these measures are certainly useful, they tell us little about the structure of the neural code, which is what we are concerned with here. Furthermore, we would also like to elucidate how this structure relates to the represented information items, i.e. we are interested in the semantic aspects of the neural code.

To explore the relationship between the representations of related items, Földiák [7] demonstrated that a sparse neural code can be interpreted as a graph (a kind of "semantic net"). In this interpretation, the neural responses are assumed to be binary (active/inactive). Each codeword can then be represented as a set of active units (a subset of all units). The codewords can now be partially ordered under set inclusion: codeword A $\leq$ codeword B iff the set of active neurons of A is a subset of the active neurons of B. This ordering relation is capable of capturing semantic relationships between the represented information items. There is a duality between the information items and the sets representing them: a more general class corresponds to a smaller subset of active neurons, and more specific items are represented by larger sets [7]. Additionally, storing codewords as sets is especially efficient for sparse codes. The resulting graphs (lattices) are an interesting representation of the relationships implicit in the code.

We would also like to be able to represent how the relationship between sets of active neurons translates into the corresponding relationship between the encoded stimuli. These observations can be formalized by the well developed branch of mathematical order theory called *Formal Concept Analysis* (FCA) [8, 9]. In FCA, data from a binary relation (or *formal context*) is represented as a concept lattice. Each concept has a set of *formal objects* as an extent and a set of *formal attributes* as an intent. In our application, the stimuli are the formal objects, and the neurons are the formal attributes. The FCA approach exploits the duality of extensional and intensional descriptions and allows to visually explore the data in lattice diagrams. FCA has shown to be useful for data exploration and knowledge discovery in numerous applications in a variety of fields [10, 11].

We give a short introduction to FCA in section 2 and demonstrate how the sparseness (or denseness) of the neural code affects the structure of the concept lattice in section 3. Section 4 describes the generative classifier model which we use to build the formal context from the responses of neurons in the high-level visual cortex of monkeys. Finally, we discuss the concept lattices so obtained in section 5.

## 2 Formal Concept Analysis

Central to FCA[9] is the notion of the formal context $K := (G, M, I)$, which is comprised of a set of formal objects $G$, a set of formal attributes $M$ and a binary relation $I \subseteq G \times M$ between members of $G$ and $M$. In our application, the members of $G$ are visual stimuli, whereas the members of $M$ are the neurons. If neuron $m \in M$ responds when stimulus $g \in G$ is presented, then we write $(g, m) \in I$ or $gIm$. It is customary to represent the context as a cross table, where the row(column) headings are the object(attribute) names. For each pair $(g, m) \in I$, the corresponding cell in the cross table has an "x". Table 1, left, shows a simple example context.

| | n1 | n2 | n3 |
|---|---|---|---|
| monkeyFace | × | × | |
| monkeyHand | | × | |
| humanFace | × | | |
| spider | | | × |

| concept | extent (stimuli) | intent (neurons) |
|---|---|---|
| 0 | *ALL* | *NONE* |
| 1 | spider | n3 |
| 2 | humanFace monkeyFace | n1 |
| 3 | monkeyFace monkeyHand | n2 |
| 4 | monkeyFace | n1 n2 |
| 5 | *NONE* | *ALL* |

Table 1: *Left*: a simple example context, represented as a cross-table. The objects (rows) are 4 visual stimuli, the attributes (columns) are 3 (hypothetical) neurons n1,n2,n3. An "x" in a cell indicates that a stimulus elicited a response from the corresponding neuron. *Right*: the concepts of this context. Concepts are lectically ordered [9]. Colors correspond to fig.1.

Define the prime operator for subsets $A \subseteq G$ as $A' = \{m \in M | \forall g \in A : gIm\}$ i.e. $A'$ is the set of all attributes shared by the objects in $A$. Likewise, for $B \subseteq M$ define $B' = \{g \in G | \forall m \in B : gIm\}$ i.e. $B'$ is the set of all objects having all attributes in $B$.

**Definition 2.1** *[9] A **formal concept** of the context $K$ is a pair $(A, B)$ with $A \subseteq G$, $B \subseteq M$ such that $A' = B$ and $B' = A$. $A$ is called the **extent** and $B$ is the **intent** of the concept $(A, B)$. $\mathbb{B}(K)$ denotes the set of all concepts of the context $K$.*

In other words, given the relation $I$, $(A, B)$ is a concept if $A$ determines $B$ and vice versa. $A$ and $B$ are sometimes called *closed* subsets of $G$ and $M$ with respect to $I$. Table 1, right, lists all concepts of the context in table 1, left. One can visualize the defining property of a concept as follows: if $(A, B)$ is a concept, reorder the rows and columns of the cross table such that all objects in $A$ are in adjacent rows, and all attributes in $B$ are in adjacent columns. The cells corresponding to all $g \in A$ and $m \in B$ then form a rectangular block of "x"s with no empty spaces in between. In the example above, this can be seen (without reordering rows and columns) for concepts 1,3,4. For a graphical representation of the relationships between concepts, one defines an order $\mathbb{B}(K)$:

**Definition 2.2** *[9] If $(A_1, B_1)$ and $(A_2, B_2)$ are concepts of a context, $(A_1, B_1)$ is a **subconcept** of $(A_2, B_2)$ if $A_1 \subseteq A_2$ (which is equivalent to $B_1 \supseteq B_2$). In this case, $(A_2, B_2)$ is a **superconcept** of $(A_1, B_1)$ and we write $(A_1, B_1) \leq (A_2, B_2)$. The relation $\leq$ is called the **order** of the concepts.*

It can be shown [8, 9] that $I\!B(K)$ and the concept order form a complete lattice. The concept lattice of the context in table 1, with full and reduced labeling, is shown in fig.1. Full labeling means that a concept node is depicted with its full extent and intent. A reduced labeled concept lattice shows an object only in the smallest (w.r.t. the concept order of definition 2.2) concept of whose extent the object is a member. This concept is called the *object concept*, or the concept that *introduces* the object. Likewise, an attribute is shown only in the largest concept of whose intent the attribute is a member, the *attribute concept*, which *introduces* the attribute. The closedness of extents and intents has an important consequence for neuroscientific applications. Adding attributes to $M$ (e.g. responses of additional neurons) will very probably grow $I\!B(K)$. However, the original concepts will be embedded as a substructure in the larger lattice, with their ordering relationships preserved.

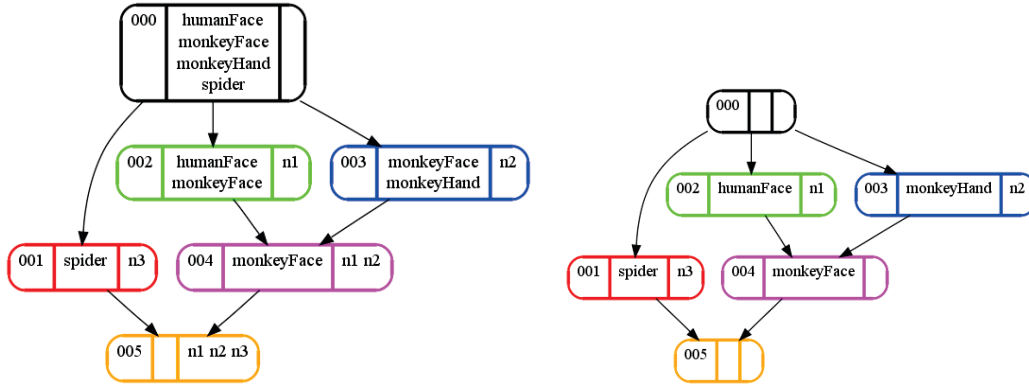

Figure 1: Concept lattice computed from the context in table 1. Each node is a concept, arrows represent superconcept relation, i.e. an arrow from $X$ to $Y$ reads: $X$ is a superconcept of $Y$. Colors correspond to table 1, right. The number in the leftmost compartment is the concept number. Middle compartment contains the extent, rightmost compartment the intent. *Left*: fully labeled concepts, i.e. all members of extents and intents are listed in each concept node. *Right*: reduced labeling. An object/attribute is only listed in the extent/intent of the smallest/largest concept that contains it. Reduced labeling is very useful for drawing large concept lattices.

The lattice diagrams make the ordering relationship between the concepts graphically explicit: concept 3 contains all "monkey-related" stimuli, concept 2 encompasses all "faces". They have a common child, concept 4, which is the "monkeyFace" concept. The "spider" concept (concept 1) is incomparable to any other concept except the top and the bottom of the lattice. Note that these relationships arise as a consequence of the (here hypothetical) response behavior of the neurons. We will show (section 5) that the response patterns of real neurons can lead to similarly interpretable structures.

From a decoding perspective, a fully labeled concept shows those stimuli that have activated at least the set of neurons in the intent. In contrast, the stimuli associated with a concept in reduced labeling will activate the set of neurons in the intent, but no others. The fully labeled concepts show stimuli encoded by activity of the active neurons of the concept without knowledge of the firing state of the other neurons. Reduced labels, on the other hand show those stimuli that elicited a response *only* from the neurons in the intent.

## 3 Concept lattices of local, sparse and dense codes

One feature of neural codes which has attracted a considerable amount of interest is its *sparseness*. In the case of a binary neural code, the sparseness of a codeword is inversely related to the fraction of active neurons. The average sparseness across all codewords is the sparseness of the code [12, 13]. Sparse codes, i.e. codes where this fraction is low, are found interesting for a variety of reasons: they offer a good compromise between encoding capacity, ease of decoding and robustness [14], they seem to be employed in the mammalian visual processing system [15] and they are well suited to representing the visual environment we live in [15, 16]. It is also possible to define sparseness for graded or even continuous-valued responses (see e.g. [17, 4, 13]). To study what structural

effects different levels of sparseness would have on a neural code, we generated random codes, i.e. each of 10 stimuli was associated with randomly drawn responses of 10 neurons, subject to the constraints that the code be perfectly decodable and that the sparseness of each codeword was equal to the sparseness of the code. Fig.2 shows the contexts (represented as cross-tables) and the concept lattices of a local code (activity ratio 0.1), a sparse code (activity ratio 0.2) and a dense code (activity ratio 0.5).

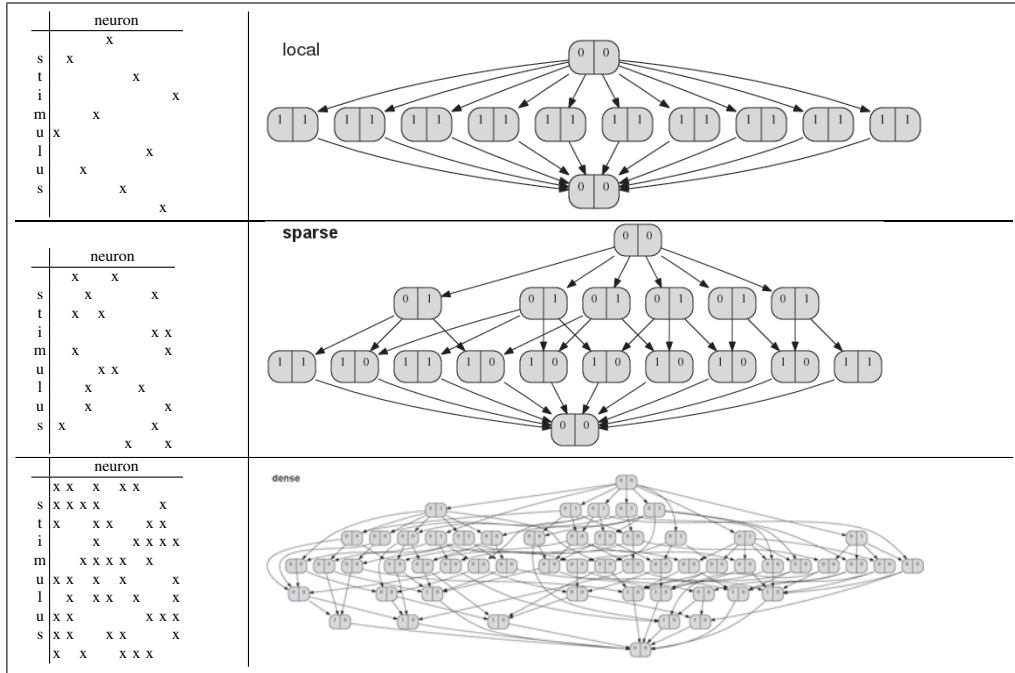

Figure 2: Contexts (represented as cross-tables) and concept lattices for a local, sparse and dense random neural code. Each context was built out of the responses of 10 (hypothetical) neurons to 10 stimuli. Each node represents a concept, the left(right) compartment contains the number of introduced stimuli(neurons). In a local code, the response patters to different stimuli have no overlapping activations, hence the lattice representing this code is an antichain with top and bottom element added. Each concept in the antichain introduces (at least) one stimulus and (at least) one neuron. In contrast, a dense code results in a lot of concepts which introduce neither a stimulus nor a neuron. The lattice of the dense code is also substantially longer than that of the sparse and local codes.

The most obvious differences between the lattices is the total number of concepts. A dense code, even for a small number of stimuli, will give rise to a lot of concepts, because the neuron sets representing the stimuli are very probably going to have non-empty intersections. These intersections are potentially the intents of concepts which are larger than those concepts that introduce the stimuli. Hence, the latter are found towards the bottom of the lattice. This implies that they have large intents, which is of course a consequence of the density of the code. Determining these intents thus requires the observation of a large number of neurons, which is unappealing from a decoding perspective. The local code does not have this drawback, but is hampered by a small encoding capacity (maximal number of concepts with non-empty extents): the concept lattice in fig.2 is the largest one which can be constructed for a local code comprised of 10 binary neurons. Which of the above structures is most appropriate depends on the conceptual structure of environment to be encoded.

## 4  Building a formal context from responses of high-level visual neurons

To explore whether FCA is a suitable tool for interpreting real neural codes, we constructed formal contexts from the responses of high-level visual cortical cells in area STSa (part of the temporal lobe) of monkeys. Characterizing the responses of these cells is a difficult task. They exhibit complex

nonlinearities and invariances which make it impossible to apply linear techniques, such as reverse correlation [18, 19, 20]. The concept lattice obtained by FCA might enable us to display and browse these invariances: if the response of a subset of cells indicates the presence of an invariant feature in a stimulus, then all stimuli having this feature should form the extent of a concept whose intent is given by the responding cells, much like the "monkey" and "face" concepts in the example in section 2.

## 4.1 Physiological data

The data were obtained through [21], where the experimental details can be found. Briefly, spike trains were obtained from neurons within the upper and lower banks of the superior temporal sulcus (STSa) via standard extracellular recording techniques [22] from an awake and behaving monkey (*Macaca mulatta*) performing a fixation task. This area contains cells which are responsive to faces. The recorded firing patters were turned into distinct samples, each of which contained the spikes from $-300$ ms before to 600 ms after the stimulus onset with a temporal resolution of 1 ms. The stimulus set consisted of 1704 images, containing color and black and white views of human and monkey head and body, animals, fruits, natural outdoor scenes, abstract drawings and cartoons. Stimuli were presented for 55ms each without inter-stimulus gaps in random sequences. While this rapid serial visual presentation (RSVP) paradigm complicates the task of extracting stimulus-related information from the spiketrains, it has the advantage of allowing for the testing of a large number of stimuli. A given cell was tested on a subset of 600 or 1200 of these stimuli, each stimulus was presented between 1-15 times.

## 4.2 Bayesian thresholding

Before we can apply FCA, we need to extract a binary attribute from the raw spiketrains. While FCA can also deal with many-valued attributes, see [23, 9], we will employ binary thresholding as a starting point. Moreover, when time windows are limited (e.g. in the RSVP condition) it is usually impossible to extract more than 1 bit of stimulus identity-related information from a spiketrain per stimulus [24]. We do not suggest that real neurons have a binary activation function. We are merely concerned with finding a maximally informative response binarization, to allow for the construction of meaningful concepts. We do this by Bayesian thresholding, as detailed in appendix A. This procedure also avails us of a null hypothesis $H_0 =$"the responses contain no information about the stimuli".

## 4.3 Cell selection

The experimental data consisted of recordings from 26 cells. To minimize the risk that the computed neural responses were a result of random fluctuations, we excluded a cell if 1.) $H_0$ was more probable than $10^{-6}$ or 2.) the posterior standard deviations of the counting window parameters were larger than $20ms$, indicating large uncertainties about the response timing. Cells which did not respond above the threshold included all cells excluded by the above criteria (except one). Furthermore, since not all cells were tested on all stimuli, we also had to select pairs of subsets of cells and stimuli such that all cells in a pair were tested on all stimuli. Incidentally, this selection can also be accomplished with FCA, by determining the concepts of a context with $gJm =$"stimulus $g$ was tested on cell $m$" and selecting those with a large number of stimuli $\times$ number of cells. Two of these cell and stimulus subset pairs ("A", containing 364 stimuli and 13 cells, and "B", containing 600 stimuli, 12 cells) were selected for further analysis.

## 5 Results

To analyze the neural code, the thresholded neural resposes were used to build stimulus-by-cell-response contexts. We performed FCA on these with COLIBRICONCEPTS[1], created stimulus image montages and plotted the lattices[2]. The complete concept lattices were too large to display on a page. Graphs of lattices A and B with reduced labeling on the stimuli are included in the supplementary

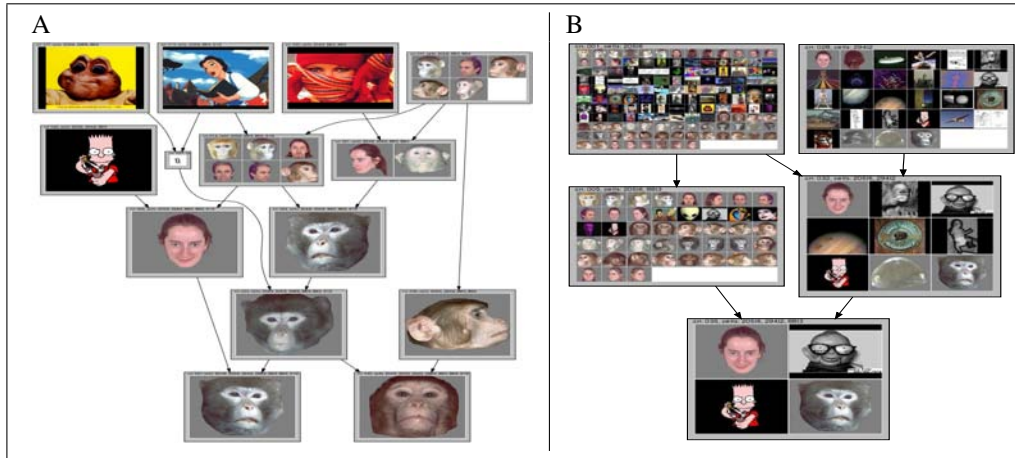

Figure 3: *A*: a subgraph of lattice A with reduced labeling on the stimuli, i.e. stimuli are only shown in their object concepts. The ∅ indicates that an extent is the intersection of its superconcepts' extents, i.e. no new stimuli were introduced by this concept. All cells forming this part of the concept lattice were responsive to faces. *B*: a subgraph of lattice B, fully labeled. The concepts on the right side are not exclusively "face" concepts, but most members of their extents have something "roundish" about them.

material (files `latticeA_neuroFCA.pdf` and `latticeB_neuroFCA.pdf`). In these graphs, the top of the frame around each concept image contains the concept number and the list of cells in the intent.

Fig.3, A shows a subgraph from lattice A, which exclusively contained "face" concepts. This subgraph, with full labeling, is also a part of the supplementary material (file `faceSubgraphLatticeA_neuroFCA.pdf`). The top concepts introduce human and cartoon faces, i.e. their extents are consist of general "face" images, while their intents are small (3 cells). In contrast, the lower concepts introduce mostly single monkey faces, with the bottom concepts having an intent of 7 cells. We may interpret this as an indication that the neural code has a higher "resolution" for faces of conspecifics than for faces in general, i.e. other monkeys are represented in greater detail in a monkey's brain than humans or cartoons. This feature can be observed in most lattices we generated.

Fig.3, B shows a subgraph from lattice B with full labeling. The concepts in the left half of the graph are face concepts, whereas the extents of the concepts in the right half also contain a number of non-face stimuli. Most of the latter have something "roundish" about them. The bottom concept, being subordinate to both the "round" and the "face" concepts, encompasses stimuli with both characteristics, which points towards a product-of-experts encoding [25]. This example also highlights another advantage of FCA over standard hierarchical analysis techniques, e.g. hierarchical clustering: it does not impose a tree structure when the data do not support it (a shortcoming of the analysis in [26]).

For preliminary validation, we experimented with stimulus shuffling (i.e. randomly assigning stimuli to the recorded responses) to determine whether the found concepts are indeed meaningful. This procedure leaves the lattice structure intact, but mixes up the extents. A 'naive' observer was then no longer able to label the concepts (as in fig.3, 'round', 'face' or 'conspecifics'). Evidence of concept stability was obtained by trying different binarization thresholds: as stated in appendix A, we used a threshold probability of 0.5. This threshold can be raised up to 0.7 without losing any of the conceptual structures described in fig.3, although some of the stimuli migrate upwards in the lattice.

# 6 Conclusion

We demonstrated the potential usefulness of FCA for the exploration and interpretation of neural codes. This technique is feasible even for high-level visual codes, where linear decoding methods [19, 20] fail, and it provides qualitative information about the structure of the code which goes beyond stimulus label decoding [4]. Clearly, this application of FCA is still in its infancy. It would be very interesting to repeat the analysis presented here on data obtained from simultaneous multi-cell recordings, to elucidate whether the conceptual structures derived by FCA are used for decoding by real brains. On a larger scale than single neurons, FCA could also be employed to study the relationships in fMRI data [27].

**Acknowledgment** D. Endres was supported by MRC fellowship G0501319.

## Footnotes

[1]see http://code.google.com/p/colibri-concepts/

[2]with IMAGEMAGICK, http://www.imagemagick.org and GRAPHVIZ, http://www.graphviz.org

## References

[1] A. P. Georgopoulos, A. B. Schwartz, and R. E. Kettner. Neuronal population coding of movement direction. *Science*, 233(4771):1416–1419, 1986.

[2] P Földiák. The 'Ideal Homunculus': Decoding neural population responses by Bayesian inference. *Perception*, 22 suppl:43, 1993.

[3] MW Oram, P Földiák, DI Perrett, and F Sengpiel. The 'Ideal Homunculus': decoding neural population signals. *Trends In Neurosciences*, 21:259–265, June 1998.

[4] R. Q. Quiroga, L. Reddy, C. Koch, and I. Fried. Decoding Visual Inputs From Multiple Neurons in the Human Temporal Lobe. *J Neurophysiol*, 98(4):1997–2007, 2007.

[5] OR Duda, PE Hart, and DG Stork. *Pattern classification*. John Wiley & Sons, New York, Chichester, 2001.

[6] T. M. Cover and J. A. Thomas. *Elements of Information Theory*. John Wiley & Sons, New York, 1991.

[7] P Földiák. Sparse neural representation for semantic indexing. In *XIII Conference of the European Society of Cognitive Psychology (ESCOP-2003)*, 2003. http://www.st-andrews.ac.uk/∼pf2/escopill2.pdf.

[8] R. Wille. Restructuring lattice theory: an approach based on hierarchies of concepts. In I. Rival, editor, *Ordered sets*, pages 445–470. Reidel, Dordrecht-Boston, 1982.

[9] Bernhard Ganter and Rudolf Wille. *Formal Concept Analysis: Mathematical foundations*. Springer, 1999.

[10] B. Ganter, G. Stumme, and R. Wille, editors. *Formal Concept Analysis, Foundations and Applications*, volume 3626 of *Lecture Notes in Computer Science*. Springer, 2005.

[11] U. Priss. Formal concept analysis in information science. *Annual Review of Information Science and Technology*, 40:521–543, 2006.

[12] P Földiák. Sparse coding in the primate cortex. In Michael A Arbib, editor, *The Handbook of Brain Theory and Neural Networks*, pages 1064–1068. MIT Press, second edition, 2002.

[13] P Földiák and D Endres. Sparse coding. *Scholarpedia*, 3(1):2984, 2008. http://www.scholarpedia.org/article/Sparse_coding.

[14] P Földiák. Forming sparse representations by local anti-Hebbian learning. *Biological Cybernetics*, 64:165–170, 1990.

[15] B. A Olshausen, D. J Field, and A Pelah. Sparse coding with an overcomplete basis set: a strategy employed by V1. *Vision Res.*, 37(23):3311–3325, 1997.

[16] Eero P Simoncelli and Bruno A Olshausen. Natural image statistics and neural representation. *Annual Review of Neuroscience*, 24:1193–1216, 2001.

[17] ET Rolls and A Treves. The relative advantages of sparse versus distributed encoding for neuronal networks in the brain. *Network*, 1:407–421, 1990.

[18] P Dayan and LF Abbott. *Theoretical Neuroscience*. MIT Press, London, Cambridge, 2001.

[19] J.P. Jones and L. A. Palmer. An evaluation of the two-dimensional Gabor filter model of simple receptive fields in cat striate cortex. *Journal of Neurophysiology*, 58(6):1233–1258, 1987.

[20] D. L. Ringach. Spatial structure and symmetry of simple-cell receptive fields in macaque primary visual cortex. *Journal of Neurophysiology*, 88:455–463, 2002.

[21] P Földiák, D Xiao, C Keysers, R Edwards, and DI Perrett. Rapid serial visual presentation for the determination of neural selectivity in area STSa. *Progress in Brain Research*, pages 107–116, 2004.

[22] M. W. Oram and D. I. Perrett. Time course of neural responses discriminating different views of the face and head. *Journal of Neurophysiology*, 68(1):70–84, 1992.

[23] R. Wille and F. Lehmann. A triadic approach to formal concept analysis. In G. Ellis, R. Levinson, W. Rich, and J. F. Sowa, editors, *Conceptual structures: applications, implementation and theory*, pages 32–43. Springer, Berlin-Heidelberg-New York, 1995.

[24] D. Endres. *Bayesian and Information-Theoretic Tools for Neuroscience*. PhD thesis, School of Psychology, University of St. Andrews, U.K., 2006. http://hdl.handle.net/10023/162.

[25] GE Hinton. Products of experts. In *Ninth International Conference on Artificial Neural Networks ICANN 99*, number 470 in ICANN, 1999.

[26] R Kiani, H Esteky, K Mirpour, and K Tanaka. Object category structure in response patterns of neuronal population in monkey inferior temporal cortex. *Journal of Neurophysiology*, 97(6):4296–4309, April 2007.

[27] K. N. Kay, T. Naselaris, R. J. Prenger, and J. L. Gallant. Identifying natural images from human brain activity. *Nature*, 452:352–255, 2008. http://dx.doi.org/10.1038/nature06713.

[28] D. Endres and P. Földiák. Exact Bayesian bin classification: a fast alternative to bayesian classification and its application to neural response analysis. *Journal of Computational Neuroscience*, 24(1):24–35, 2008. DOI: 10.1007/s10827-007-0039-5.

## A    Method of Bayesian thresholding

A standard way of obtaining binary responses from neurons is thresholding the spike count within a certain time window. This is a relatively straightforward task, if the stimuli are presented well separated in time and a lot of trials per stimulus are available. Then latencies and response offsets are often clearly discernible and thus choosing the time window is not too difficult. However, under RSVP conditions with few trials per stimulus, response separation becomes more tricky, as the responses to subsequent stimuli will tend to follow each other without an intermediate return to baseline activity. Moreover, neural responses tend to be rather noisy. We will therefore employ a simplified version of the generative Bayesian Bin classification algorithm (BBCa) [28], which was shown to perform well on RSVP data [24].

BBCa was designed for the purpose of inferring stimulus labels $g$ from a continuous-valued, scalar measure $z$ of a neural response. The range of $z$ is divided into a number of contiguous bins. Within each bin, the observation model for the $g$ is a Bernoulli scheme with a Dirichlet prior over its parameters. It is shown in [28] that one can iterate/integrate over all possible bin boundary configurations efficiently, thus making exact Bayesian inference feasible. We make two simplifications to BBCa: 1) $z$ is discrete, because we are counting spikes and 2) we use models with only 1 bin boundary in the range of $z$. The bin membership of a given neural response can then serve as the binary attribute required for FCA, since BBCa weighs bin configurations by their classification (i.e. stimulus label decoding) performance. We proceed in a straight Bayesian fashion: since the bin membership is the only variable we are interested in, all other parameters (counting window size and position, class membership probabilities, bin boundaries) are marginalized. This minimizes the risk of spurious results due to "contrived" information (i.e. choices of parameters) made at some stage of the inference process. Afterwards, the probability that the response belongs to the upper bin is thresholded at a probability of 0.5. BBCa can also be used for model comparison. Running the algorithm with no bin boundaries in the range of $z$ effectively yields the probability of the data given the "null hypothesis" $H_0$: $z$ does not contain any information about $g$. We can then compare it against the alternative hypothesis described above (i.e. the information which bin $z$ is in tells us something about $g$) to determine whether the cell has responded at all.